# Information through a Spiking Neuron

**Charles F. Stevens and Anthony Zador**
Salk Institute MNL/S
La Jolla, CA 92037
zador@salk.edu

## Abstract

While it is generally agreed that neurons transmit information about their synaptic inputs through spike trains, the code by which this information is transmitted is not well understood. An upper bound on the information encoded is obtained by hypothesizing that the precise timing of each spike conveys information. Here we develop a general approach to quantifying the information carried by spike trains under this hypothesis, and apply it to the leaky integrate-and-fire (IF) model of neuronal dynamics. We formulate the problem in terms of the probability distribution $p(T)$ of interspike intervals (ISIs), assuming that spikes are detected with arbitrary but finite temporal resolution. In the absence of added noise, all the variability in the ISIs could encode information, and the information rate is simply the entropy of the ISI distribution, $H(T) = \langle -p(T) \log_2 p(T) \rangle$, times the spike rate. $H(T)$ thus provides an *exact* expression for the information rate. The methods developed here can be used to determine experimentally the information carried by spike trains, even when the lower bound of the information rate provided by the stimulus reconstruction method is not tight. In a preliminary series of experiments, we have used these methods to estimate information rates of hippocampal neurons in slice in response to somatic current injection. These pilot experiments suggest information rates as high as 6.3 bits/spike.

## 1 Information rate of spike trains

Cortical neurons use spike trains to communicate with other neurons. The output of each neuron is a stochastic function of its input from the other neurons. It is of interest to know how much each neuron is telling other neurons about its inputs.

How much information does the spike train provide about a signal? Consider noise $n(t)$ added to a signal $s(t)$ to produce some total input $y(t) = s(t) + n(t)$. This is then passed through a (possibly stochastic) functional $\mathcal{F}$ to produce the output spike train $\mathcal{F}[y(t)] \rightarrow z(t)$. We assume that *all* the information contained in the spike train can be represented by the list of spike times; that is, there is no extra information contained in properties such as spike height or width. Note, however, that many characteristics of the spike train such as the mean or instantaneous rate

can be derived from this representation; if such a derivative property turns out to be the relevant one, then this formulation can be specialized appropriately.

We will be interested, then, in the mutual information $I(S(t); Z(t))$ between the input signal ensemble $S(t)$ and the output spike train ensemble $Z(t)$. This is defined in terms of the entropy $H(S)$ of the signal, the entropy $H(Z)$ of the spike train, and their joint entropy $H(S, Z)$,

$$I(S; Z) = H(S) + H(Z) - H(S, Z). \tag{1}$$

Note that the mutual information is symmetric, $I(S; Z) = I(Z; S)$, since the joint entropy $H(S, Z) = H(Z, S)$. Note also that if the signal $S(t)$ and the spike train $Z(t)$ are completely independent, then the mutual information is 0, since the joint entropy is just the sum of the individual entropies $H(S, Z) = H(S) + H(Z)$. This is completely in line with our intuition, since in this case the spike train can provide no information about the signal.

## 1.1   Information estimation through stimulus reconstruction

Bialek and colleagues (Bialek et al., 1991) have used the *reconstruction method* to obtain a strict lower bound on the mutual information in an experimental setting. This method is based on an expression mathematically equivalent to eq. (1) involving the conditional entropy $H(S|Z)$ of the signal given the spike train,

$$\begin{aligned} I(S; Z) &= H(S) - H(S|Z) \\ &\geq H(S) - H_{\text{est}}(S|Z), \end{aligned} \tag{2}$$

where $H_{\text{est}}(S|Z)$ is an upper bound on the conditional entropy obtained from a reconstruction $s_{\text{est}}(t)$ of the signal. The entropy is estimated from the second order statistics of the reconstruction error $e(t) \stackrel{\triangle}{=} s(t) - s_{\text{est}}(t)$; from the maximum entropy property of the Gaussian this is an upper bound. Intuitively, the first equation says that the information gained about the spike train by observing the stimulus is just the initial uncertainty of the signal (in the absence of knowledge of the spike train) minus the uncertainty that remains about the signal once the spike train is known, and the second equation says that this second uncertainty must be greater for any particular estimate than for the optimal estimate.

## 1.2   Information estimation through spike train reliability

We have adopted a different approach based an equivalent expression for the mutual information:

$$I(S; Z) = H(Z) - H(Z|S). \tag{3}$$

The first term $H(Z)$ is the entropy of the spike train, while the second $H(Z|S)$ is the conditional entropy of the spike train given the signal; intuitively this like the inverse *repeatability* of the spike train given repeated applications of the same signal. Eq. (3) has the advantage that, if the spike train is a deterministic function of the input, it permits *exact* calculation of the mutual information. This follows from an important difference between the conditional entropy term here and in eq. 2: whereas $H(S|Z)$ has both a deterministic and a stochastic component, $H(Z|S)$ has only a stochastic component. Thus in the absence of added noise, the discrete entropy $H(Z|S) = 0$, and eq. (3) reduces to $I(S; Z) = H(Z)$.

If ISIs are independent, then the $H(Z)$ can be simply expressed in terms of the entropy of the (discrete) ISI distribution $p(T)$,

$$H(T) = -\sum_{i=0}^{\infty} p(T_i) \log_2 p(T_i) \tag{4}$$

as $H(Z) = nH(T)$, where $n$ is the number of spikes in $Z$. Here $p(T_i)$ is the probability that the spike occurred in the interval $(i)\Delta t$ to $(i + 1)\Delta t$. The assumption of finite timing precision $\Delta t$ keeps the potential information finite. The advantage of considering the ISI distribution $p(T)$ rather than the full spike train distribution $p(Z)$ is that the former is univariate while the latter is multivariate; estimating the former requires much less data.

Under what conditions are ISIs independent? Correlations between ISIs can arise either through the stimulus or the spike generation mechanism itself. Below we shall guarantee that correlations do not arise from the spike-generator by considering the *forgetful integrate-and-fire* (IF) model, in which all information about the previous spike is eliminated by the next spike. If we further limit ourselves to temporally uncorrelated stimuli (*i.e.* stimuli drawn from a white noise ensemble), then we can be sure that ISIs are independent, and eq. (4) can be applied.

In the presence of noise, $H(Z|T)$ must also be evaluated, to give

$$I(S;T) = H(T) - H(T|S). \tag{5}$$

$H(T|S)$ is the conditional entropy of the ISI given the signal,

$$H(T|S) = -\left\langle \sum_{j=1}^{\infty} p(T_j|s_i(t)) \log_2 p(T_j|s_i(t)) \right\rangle_{s_i(t)} \tag{6}$$

where $p(T_j|s_i(t))$ is the probability of obtaining an ISI of $T_j$ in response to a particular stimulus $s_i(t)$ in the presence of noise $n(t)$. The conditional entropy can be thought of as a quantification of the reliability of the spike generating mechanism: it is the average trial-to-trial variability of the spike train generated in response to repeated applications of the same stimulus.

### 1.3   Maximum spike train entropy

In what follows, it will be useful to compare the information rate for the IF neuron with the limiting case of an exponential ISI distribution, which has the maximum entropy for any point process of the given rate (Papoulis, 1984). This provides an upper bound on the information rate possible for any spike train, given the spike rate and the temporal precision. Let $f(T) = \bar{r}e^{-\bar{r}T}$ be an exponential distribution with a mean spike rate $\bar{r}$. Assuming a temporal precision of $\Delta t$, the entropy/spike is $H(T) = \log_2 \frac{e}{\bar{r}\Delta t}$, and the entropy/time for a rate $\bar{r}$ is $\bar{r}H(T) = \bar{r}\log_2 \frac{e}{\bar{r}\Delta t}$. For example, if $\bar{r} = 1$ Hz and $\Delta t = 0.001$ sec, this gives (11.4 bits/second) (1 spike/second) = 11.4 bits/spike. That is, if we discretize a 1 Hz spike train into 1 msec bins, it is not possible for it to transmit more than 11.4 bits/second. If we reduce the bin size two-fold, the rate increases by $\log_2 1/2 = 1$ bit/spike to 12.4 bits/spike, while if we double it we lose one bit/s to get 10.4 bit/s. Note that at a different firing rate, e.g. $\bar{r} = 2$ Hz, halving the bin size still increases the entropy/spike by 1 bit/spike, but because the spike rate is twice as high, this becomes a 2 bit/second increase in the information rate.

### 1.4   The IF model

Now we consider the functional $\mathcal{F}$ describing the forgetful leaky IF model of spike generation. Suppose we add some noise $n(t)$ to a signal $s(t)$, $y(t) = n(t) + s(t)$, and threshold the sum to produce a spike train $z(t) = \mathcal{F}[s(t) + n(t)]$. Specifically, suppose the voltage $v(t)$ of the neuron obeys $\dot{v}(t) = -v(t)/\tau + y(t)$, where $\tau$ is the membrane time constant, both $s(t)$ and $n(t)$ have a white Gaussian distributions and $y(t)$ has mean $\mu$ and variance $\sigma^2$. If the voltage reaches the threshold $\theta_0$ at some time $t$, the neuron emits a spike at that time and resets to the initial condition $v_0$.

In the language of neurobiology, this model can be thought of (Tuckwell, 1988) as
the limiting case of a neuron with a leaky IF spike generating mechanism receiving
many excitatory and inhibitory synaptic inputs. Note that since the input $y(t)$ is
white, there are no correlations in the spike train induced by the signal, and *since
the neuron resets after each spike there are no correlations induced by the spike-
generating mechanism.* Thus ISIs are independent, and eq. (4) can be applied.

We will estimate the mutual information $I(S, Z)$ between the ensemble of input
signals $S$ and the ensemble of outputs $Z$. Since in this model ISIs are independent by
construction, we need only evaluate $H(T)$ and $H(T|S)$; for this we must determine
$p(T)$, the distribution of ISIs, and $p(T|s_i)$, the conditional distribution of ISIs for
an ensemble of signals $s_i(t)$. Note that $p(T)$ corresponds to the first passage time
distribution of the Ornstein-Uhlenbeck process (Tuckwell, 1988).

The neuron model we are considering has two regimes determined by the relation
of the asymptotic membrane potential (in the absence of threshold) $\mu\tau$ and the
threshold $\theta$. In the *suprathreshold* regime, $\mu\tau > \theta$, threshold crossings occur even if
the signal variance is zero ($\sigma^2 = 0$). In the *subthreshold* regime, $\mu\tau \leq \theta$, threshold
crossings occur only if $\sigma^2 > 0$. However, in the limit that $E\{T\} \gg \tau$, *i.e.* the mean
firing rate is low compared with the integration time constant (this can only occur
in the subthreshold regime), the ISI distribution is exponential, and its coefficient
of variation (CV) is unity (cf. (Softky and Koch, 1993)). In this low-rate regime the
firing is *deterministically Poisson*; by this we mean to distinguish it from the more
usual usage of *Poisson neuron*, the stochastic situation in which the instantaneous
firing rate parameter (the probability of firing over some interval) depends on the
stimulus (*i.e.* $\bar{r} \propto s(t)$). In the present case the exponential ISI distribution arises
from a deterministic mechanism.

At the border between these regimes, when the threshold is just equal to the asymp-
totic potential, $\theta_0 = \mu\tau$, we have an explicit and exact solution for the entire ISI
distribution (Sugiyama et al., 1970)

$$p(T) = \frac{(\mu\tau)(\tau/2)^{-3/2}}{(2\pi)^{1/2}\sigma}[e^{2T/\tau} - 1]^{-3/2}\exp(2T/\tau - \frac{(\mu\tau)^2}{(\sigma^2\tau)(e^{2T/\tau} - 1)}). \quad (7)$$

This is the special case where, in the absence of fluctuations ($\sigma^2 = 0$), the membrane
potential hovers just subthreshold. Its neurophysiological interpretation is that the
excitatory inputs just balance the inhibitory inputs, so that the neuron hovers just
on the verge of firing.

## 1.5 Information rates for noisy and noiseless signals

Here we compare the information rate for a IF neuron at the "balance point" $\mu\tau = \theta$
with the maximum entropy spike train. For simplicity and brevity we consider only
the zero-noise case, *i.e.* $n(t) = 0$. Fig. 1A shows the information per spike as a
function of the firing rate calculated from eq. (7), which was varied by changing
the signal variance $\sigma^2$. We assume that spikes can be resolved with a temporal
resolution of 1 msec, *i.e.* that the ISI distribution has bins 1 msec wide. The
*dashed* line shows the theoretical upper bound given by the exponential distribution;
this limit can be approached by a neuron operating far below threshold, in the
Poisson limit. For both the IF model and the upper bound, the information per
spike is a monotonically decreasing function of the spike rate; the model almost
achieves the upper bound when the mean ISI is just equal to the membrane time
constant. In the model the information saturates at very low firing rates, but for the
exponential distribution the information increases without bound. At high firing
rates the information goes to zero when the firing rate is too fast for individual ISIs
to be resolved at the temporal resolution. Fig. 1B shows that the information rate
(information per second) when the neuron is at the balance point goes through a

maximum as the firing rate increases. The maximum occurs at a lower firing rate than for the exponential distribution (*dashed line*).

### 1.6    Bounding information rates by stimulus reconstruction

By construction, eq. (3) gives an exact expression for the information rate in this model. We can therefore compare the lower bound provided by the stimulus reconstruction method eq. (2) (Bialek et al., 1991). That is, we can assess how tight a lower bound it provides. Fig. 2 shows the lower bound provided by the reconstruction (*solid line*) and the reliability (*dashed line*) methods as a function of the firing rate. The firing rate was increased by increasing the mean $\mu$ of the input stimulus $y(t)$, and noise was set to 0. At low firing rates the two estimates are nearly identical, but at high firing rates the reconstruction method substantially underestimates the information rate. The amount of the underestimate depends on the model parameters, and decreases as noise is added to the stimulus. The tightness of the bound is therefore an empirical question. While Bialek and colleagues (1996) show that under the conditions of their experiments the underestimate is less than a factor of two, it is clear that the potential for underestimate under different conditions or in different systems is greater.

## 2    Discussion

While it is generally agreed that spike trains encode information about a neuron's inputs, it is not clear how that information is encoded. One idea is that it is the mean firing rate alone that encodes the signal, and that variability about this mean is effectively noise. An alternative view is that it is the variability itself that encodes the signal, *i.e.* that the information is encoded in the precise times at which spikes occur. In this view the information can be expressed in terms of the interspike interval (ISI) distribution of the spike train. This encoding scheme yields much higher information rates than one in which only the mean rate (over some interval longer than the typical ISI) is considered. Here we have quantified the information content of spike trains under the latter hypothesis for a simple neuronal model.

We consider a model in which by construction the ISIs are independent, so that the information rate (in bits/sec) can be computed directly from the information per spike (in bits/spike) and the spike rate (in spikes/sec). The information per spike in turn depends on the temporal precision with which spikes can be resolved (if precision were infinite, then the information content would be infinite as well, since any message could for example be encoded in the decimal expansion of the precise arrival time of a single spike), the reliability of the spike transduction mechanism, and the entropy of the ISI distribution itself. For low firing rates, when the neuron is in the subthreshold limit, the ISI distribution is close to the theoretically maximal exponential distribution.

Much of the recent interest in information theoretic analyses of the neural code can attributed to the seminal work of Bialek and colleagues (Bialek et al., 1991; Rieke et al., 1996), who measured the information rate for sensory neurons in a number of systems. The present results are in broad agreement with those of DeWeese (1996), who considered the information rate of a linear-filtered threshold crossing[1] (LFTC) model. DeWeese developed a functional expansion, in which the first term describes the limit in which spike times (not ISIs) are independent, and the second term is a correction for correlations. The LFTC model differs from the present IF model mainly in that it does not "reset" after each spike. Consequently the "natural"

representation of the spike train in the LFTC model is as a sequence $t_0 \ldots t_n$ of firing times, while in the IF model the "natural" representation is as a sequence $T_1 \ldots T_n$ of ISIs. The choice is one of convenience, since the two representations are equivalent.

The two models are complementary. In the LFTC model, results can be obtained for colored signals and noise, while such conditions are awkward in the IF model. In the IF model by contrast, a class of highly correlated spike trains can be conveniently considered that are awkward in the LFTC model. That is, the indendent-ISI condition required in the IF model is less restrictive than the independent-spike condition of the LFTC model—spikes are independent iff ISIs are indepenndent *and* the ISI distribution $p(T)$ is exponential. In particular, at high firing rates the ISI distribution can be far from exponential (and therefore the spikes far from independent) even when the ISIs themselves are independent.

Because we have assumed that the input $s(t)$ is white, its entropy is infinite, and the mutual information can grow without bound as the temporal precision with which spikes are resolved improves. Nevertheless, the spike train is transmitting only a minute fraction of the total available information. The signal thereby saturates the capacity of the spike train. While it is not at all clear whether this is how real neurons actually behave, it is not implausible: a typical cortical neuron receives as many as $10^4$ synaptic inputs, and if the information rate of each input is the same as the target, then the information rate impinging upon the target is $10^4$-fold greater (neglecting synaptic unreliability, which could decrease this substantially) than its capacity.

In a preliminary series of experiments, we have used the reliability method to estimate the information rate of hippocampal neuronal spike trains in slice in response to somatic current injection (Stevens and Zador, *unpublished*). Under these conditions ISIs appear to be independent, so the method developed here can be applied. In these pilot experiments, an information rates as high as 6.3 bits/spike was observed.

## Footnotes

[1]In the LFTC model, Gaussian signal and noise are convolved with a linear filter; the times at which the resulting waveform crosses some threshold are called "spikes".

# References

Bialek, W., Rieke, F., de Ruyter van Steveninck, R., and Warland, D. (1991). Reading a neural code. *Science*, 252:1854–1857.

DeWeese, M. (1996). Optimization principles for the neural code. In Hasselmo, M., editor, *Advances in Neural Information Processing Systems, vol. 8*. MIT Press, Cambridge, MA.

Papoulis, A. (1984). *Probability, random variables and stochastic processes, $2^{nd}$ edition*. McGraw-Hill.

Rieke, F., Warland, D., de Ruyter van Steveninck, R., and Bialek, W. (1996). *Neural Coding*. MIT Press.

Softky, W. and Koch, C. (1993). The highly irregular firing of cortical cells is inconsistent with temporal integration of random epsps. *J. Neuroscience.*, 13:334–350.

Sugiyama, H., Moore, G., and Perkel, D. (1970). Solutions for a stochastic model of neuronal spike production. *Mathematical Biosciences*, 8:323–341.

Tuckwell, H. (1988). *Introduction to theoretical neurobiology (2 vols.)*. Cambridge.

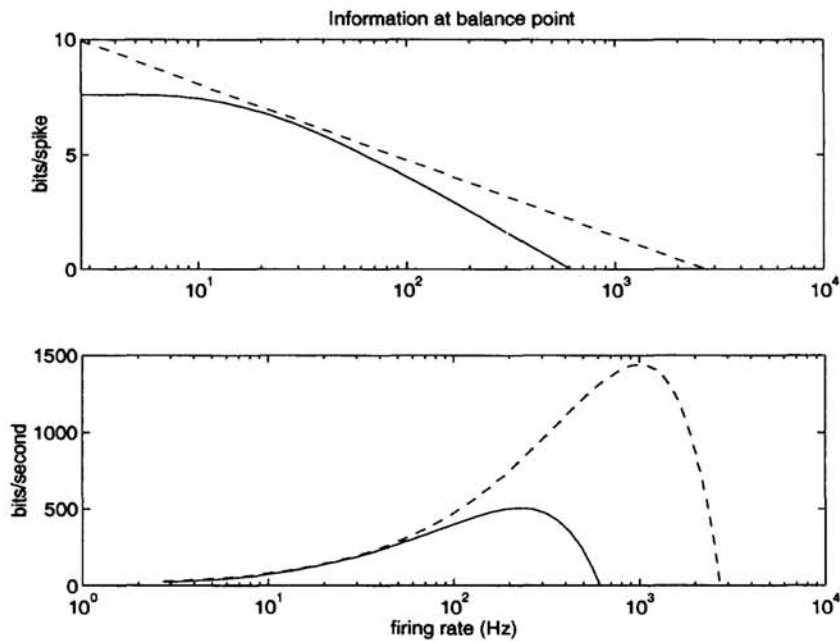

Figure 1: Information rate at balance point. (A; *top*) The information per spike decreases monotonically with the spike rate (*solid line*). It is bounded above by the entropy of the exponential limit (*dashed line*), which is the highest entropy ISI distribution for a given mean rate; this limit is approached for the IF neuron in the subthreshold regime. The information rate goes to 0 when the firing rate is of the same order as the temporal resolution $\Delta t$. The information per spike at the balance point is nearly optimal when $E\{T\} \approx \tau$. ($\tau = 50$ msec; $\Delta t = 1$ msec); (B; *bottom*) Information per second for above conditions. The information rate for both the balance point (*solid curve*) and the exponential distribution (*dashed curve*) pass through a maximum, but the maximum is greater and occurs at an higher rate for the latter. For firing rates much smaller than $\tau$, the rates are almost indistinguishable. ($\tau = 50$ msec; $\Delta t = 1$ msec)

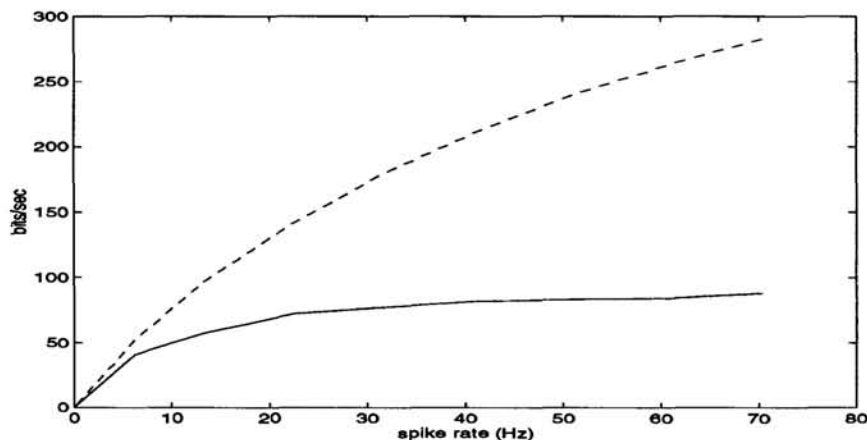

Figure 2: Estimating information by stimulus reconstruction. The information rate estimated by the reconstruction method *solid line* and the exact information rate *dashed line* are shown as a function of the firing rate. The reconstruction method significantly underestimates the actual information, particularly at high firing rates. The firing rate was varied through the mean input $\mu$. The parameters were: membrane time constant $\tau = 20$ msec; spike bin size $\Delta t = 1$ msec; signal variance $\sigma_s^2 = 0.8$; threshold $Q = 10$.